# Relative Loss Bounds for Multidimensional Regression Problems

**Jyrki Kivinen**
Department of Computer Science
P.O. Box 26 (Teollisuuskatu 23)
FIN-00014 University of Helsinki, Finland

**Manfred K. Warmuth**
Department of Computer Science
University of California, Santa Cruz
Santa Cruz, CA 95064, USA

## Abstract

We study on-line generalized linear regression with multidimensional outputs, i.e., neural networks with multiple output nodes but no hidden nodes. We allow at the final layer transfer functions such as the softmax function that need to consider the linear activations to all the output neurons. We use distance functions of a certain kind in two completely independent roles in deriving and analyzing on-line learning algorithms for such tasks. We use one distance function to define a matching loss function for the (possibly multidimensional) transfer function, which allows us to generalize earlier results from one-dimensional to multidimensional outputs. We use another distance function as a tool for measuring progress made by the on-line updates. This shows how previously studied algorithms such as gradient descent and exponentiated gradient fit into a common framework. We evaluate the performance of the algorithms using relative loss bounds that compare the loss of the on-line algoritm to the best off-line predictor from the relevant model class, thus completely eliminating probabilistic assumptions about the data.

## 1 INTRODUCTION

In a regression problem, we have a sequence of $n$-dimensional real valued *inputs* $x_t \in \mathbf{R}^n$, $t = 1, \ldots, \ell$, and for each input $x_t$ a $k$-dimensional real-valued *desired output* $y_t \in \mathbf{R}^k$. Our goal is to find a mapping that at least approximately models the dependency between $x_t$ and $y_t$. Here we consider the parametric case $\widehat{y}_t = f(\omega; x_t)$ where the *actual output* $\widehat{y}_t$ corresponding to the input $x_t$ is determined by a parameter vector $\omega \in \mathbf{R}^m$ (e.g., weights in a neural network) through a given fixed model $f$ (e.g., a neural network architecture).

Thus, we wish to obtain parameters $\omega$ such that, in some sense, $f(\omega; x_t) \approx y_t$ for all $t$. The most basic model $f$ to consider is the linear one, which in the one-dimensional case $k = 1$ means that $f(\omega; x_t) = \omega \cdot x_t$ for $\omega \in \mathbf{R}^n$. In the multidimensional case we actually have a whole matrix $\Omega \in \mathbf{R}^{k \times n}$ of parameters and $f(\Omega; x_t) = \Omega x_t$. The goodness of the fit is quantitatively measured in terms of a *loss function*; the *square loss* given by $\sum_{t,j} (y_{t,j} - \hat{y}_{t,j})^2 / 2$ is a popular choice.

In *generalized linear regression* [MN89] we fix a *transfer function* $\phi$ and apply it on top of a linear model. Thus, in the one-dimensional case we would have $f(\omega; x_t) = \phi(\omega \cdot x_t)$. Here $\phi$ is usually a continuous increasing function from $\mathbf{R}$ to $\mathbf{R}$, such as the logistic function that maps $z$ to $1/(1 + e^{-z})$. It is still possible to use the square loss, but this can lead to problems. In particular, when we apply the logistic transfer function and try to find a weight vector $\omega$ that minimizes the total square loss over $\ell$ examples $(x_t, y_t)$, we may have up to $\ell^n$ local minima [AHW95, Bud93]. Hence, some other choice of loss function might be more convenient. In the one-dimensional case it can be shown that any continuous strictly increasing transfer function $\phi$ has a specific *matching loss function* $L_\phi$ such that, among other useful properties, $\sum_t L_\phi(y_t, \phi(\omega \cdot x_t))$ is always convex in $\omega$, so local minima are not a problem [AHW95]. For example, the matching loss function for the logistic transfer function is the *relative entropy* (a generalization of the logarithmic loss for continuous-valued outcomes). The square loss is the matching loss function for the identity transfer function (i.e., linear regression).

The main theme of the present paper is the application of a particular kind of *distance functions* to analyzing learning algorithms in (possibly multidimensional) generalized linear regression problems. We consider a particular manner in which a mapping $\phi: \mathbf{R}^k \to \mathbf{R}^k$ can be used to define a distance function $\Delta_\phi : \mathbf{R}^k \times \mathbf{R}^k \to \mathbf{R}$; the assumption we must make here is that $\phi$ has a convex potential function. The matching loss function $L_\phi$ mentioned above for a transfer function $\phi$ in the one-dimensional case is given in terms of the distance function $\Delta_\phi$ as $L_\phi(\phi(a), \phi(\hat{a})) = \Delta_\phi(\hat{a}, a)$. Here, as whenever we use the matching loss $L_\phi(y, \hat{y})$, we assume that $y$ and $\hat{y}$ are in the range of $\phi$, so we can write $y = \phi(a)$ and $\hat{y} = \phi(\hat{a})$ for some $a$ and $\hat{a}$. Notice that for $k = 1$, any strictly increasing continuous function has a convex potential (i.e., integral) function. In the more interesting case $k > 1$, we can consider transfer functions such as the softmax function, which is commonly used to transfer arbitrary vectors $a \in \mathbf{R}^k$ into probability vectors $\hat{y}$ (i.e., vectors such that $\hat{y}_i \geq 0$ for all $i$ and $\sum_i \hat{y}_i = 1$). The matching loss function for the softmax function defined analogously with the one-dimensional case turns out to be the relative entropy (or Kullback-Leibler divergence), which indeed is a commonly used measure of distance between probability vectors. For the identity transfer function, the matching loss function is the squared Euclidean distance.

The first result we get from this observation connecting matching losses to a general notion of distance is that certain previous results on generalized linear regression with matching loss on one-dimensional outputs [HKW95] directly generalize to multidimensional outputs. From a more general point of view, a much more interesting feature of these distance functions is how they allow us to view certain previously known learning algorithms, and introduce new ones, in a simple unified framework. To briefly explain this framework without unnecessary complications, we restrict the following discussion to the case $k = 1$, i.e., $f(\omega; x) = \phi(\omega \cdot x) \in \mathbf{R}$ with $\omega \in \mathbf{R}^n$.

We consider on-line learning algorithms, by which we here mean an algorithm that processes the training examples one by one, the pair $(x_t, y_t)$ being processed at time $t$. Based

on the training examples the algorithm produces a whole sequence of weight vectors $\omega_t$, $t = 1, \ldots, \ell$. At each time $t$ the old weight vector $\omega_t$ is *updated* into $\omega_{t+1}$ based on $x_t$ and $y_t$. The best-known such algorithm is on-line gradient descent. To see some alternatives, consider first a distance function $\Delta_\psi$, defined on $\mathbf{R}^n$ by some function $\psi: \mathbf{R}^n \to \mathbf{R}^n$. (Thus, we assume that $\psi$ has a convex potential.) We represent the update somewhat indirectly by introducing a new parameter vector $\theta_t \in \mathbf{R}^n$ from which the actual weights $\omega_t$ are obtained by the mapping $\omega_t = \psi(\theta_t)$. The new parameters are updated by

$$\theta_{t+1} = \theta_t - \eta \nabla_\omega \left( L_\phi(y_t, \phi(\omega \cdot x_t)) \right)_{\omega = \psi(\theta_t)} \tag{1}$$

where $\eta > 0$ is a learning rate. We call this algorithm the *general additive algorithm* with *parameterization function* $\psi$. Notice that here $\theta$ is updated by the gradient with respect to $\omega$, so this is not just a gradient descent with reparameterization [JW98]. However, we obtain the usual on-line gradient descent when $\psi$ is the identity function. When $\psi$ is the softmax function, we get the so-called exponentiated gradient (EG) algorithm [KW97, HKW95].

The connection of the distance function $\Delta_\psi$, to the update (1) is two-fold. First, (1) can be motivated as an approximate solution to a minimization problem in which the distance $\Delta_\psi(\theta_t, \theta_{t+1})$ is used as a kind of penalty term to prevent too drastic an update based on a single example. Second, the distance function $\Delta_\psi$, can be used as a potential function in analyzing the performance of the resulting algorithm. The same distance functions have been used previously for exactly the same purposes [KW97, HKW95] in important special cases (the gradient descent and EG algorithms) but without realizing the full generality of the method.

It should be noted that the choice of the parameterization function $\psi$ is left completely free, as long as $\psi$ has a convex potential function. (In contrast, the choice of the transfer function $\phi$ depends on what kind of a regression problem we wish to solve.) Earlier work suggests that the softmax parameterization function (i.e., the EG algorithm) is particularly suited for situations in which some sparse weight vector $\omega$ gives a good match to the data [HKW95, KW97]. (Because softmax normalizes the weight vector and makes the components positive, a simple transformation of the input data is typically added to realize positive and negative weights with arbitrary norm.)

In work parallel to this, the analogue of the general additive update (1) in the context of linear classification, i.e., with a threshold transfer function, has recently been developed and analyzed by Grove et al. [GLS97] with methods and results very similar to ours. Cesa-Bianchi [CB97] has used somewhat different methods to obtain bounds also in cases in which the loss function does not match the transfer function. Jagota and Warmuth [JW98] view (1) as an Euler discretization of a system of partial differential equations and investigate the performance of the algorithm as the discretization parameter approaches zero.

The distance functions we use here have previously been applied in the context of exponential families by Amari [Ama85] and others. Here we only need some basic technical properties of the distance functions that can easily be derived from the definitions. For a discussion of our line of work in a statistical context see Azoury and Warmuth [AW97].

In Section 2 we review the definition of a matching loss function and give examples. Section 3 discusses the general additive algorithm in more detail. The actual relative on-line loss bounds we have for the general additive algorithm are explained in Section 4.

## 2   DISTANCE FUNCTIONS AND MATCHING LOSSES

Let $\phi\colon \mathbf{R}^k \to \mathbf{R}^k$ be a function that has a convex potential function $P_\phi$ (i.e., $\phi = \nabla P_\phi$ for some convex $P_\phi\colon \mathbf{R}^k \to \mathbf{R}$). We first define a *distance function* $\Delta_\phi$ for $\phi$ by

$$\Delta_\phi(\widehat{a}, a) = P_\phi(\widehat{a}) - P_\phi(a) + \phi(a) \cdot (a - \widehat{a}) \ . \tag{2}$$

Thus, the distance $\Delta_\phi(\widehat{a}, a)$ is the error we make if we approximate $P_\phi(\widehat{a})$ by its first-order Taylor polynomial around $a$. Convexity of $P_\phi$ implies that $\Delta_\phi$ is convex in its first argument. Further, $\Delta_\phi(\widehat{a}, a)$ is nonnegative, and zero if and only if $\phi(\widehat{a}) = \phi(a)$.

We can alternatively write (2) as $\Delta_\phi(\widehat{a}, a) = \int_a^{\widehat{a}} (\phi(r) - \phi(a)) \cdot dr$ where the integral is a path integral the value of which must be independent of the actual path chosen between $a$ and $\widehat{a}$. In the one-dimensional case, the integral is a simple definite integral, and $\phi$ has a convex potential (i.e., integral) function if it is strictly increasing and continuous [AHW95, HKW95].

Let now $\phi$ have range $V_\phi \subseteq \mathbf{R}^k$ and distance function $\Delta_\phi$. Assuming that there is a function $L_\phi\colon V_\phi \times V_\phi \to \mathbf{R}$ such that $L_\phi(\phi(a), \phi(\widehat{a})) = \Delta_\phi(\widehat{a}, a)$ holds for all $\widehat{a}$ and $a$, we say that $L_\phi$ is the *matching loss function* for $\phi$.

**Example 1** Let $\phi$ be a linear function given by $\phi(a) = Aa$ where $A \in \mathbf{R}^{k \times k}$ is symmetrical and positive definite. Then $\phi$ has the convex potential function $P_\phi(a) = a^{\mathrm{T}} A a / 2$, and (2) gives $\Delta_\phi(\widehat{a}, a) = \frac{1}{2}(a - \widehat{a})^{\mathrm{T}} A (a - \widehat{a})$. Hence, $L_\phi(y, \widehat{y}) = \frac{1}{2}(y - \widehat{y})^{\mathrm{T}} A^{-1}(y - \widehat{y})$ for all $y, \widehat{y} \in \mathbf{R}^k$.                                                                    □

**Example 2** Let $\sigma\colon \mathbf{R}^k \to \mathbf{R}^k$, $\sigma_i(a) = \exp(a_i)/\sum_{j=1}^k \exp(a_j)$, be the softmax function. It has a potential function given by $P_\sigma(a) = \ln \sum_{j=1}^k \exp(a_j)$. To see that $P_\sigma$ is convex, notice that the Hessian $\mathrm{D}^2 P_\sigma$ is given by $\mathrm{D}^2 P_\sigma(a)_{ij} = \delta_{ij}\sigma_i(a) - \sigma_i(a)\sigma_j(a)$. Given a vector $x \in \mathbf{R}^k$, let now $X$ be a random variable that has probability $\sigma_i(a)$ of taking the value $x_i$. We have $x^{\mathrm{T}} \mathrm{D}\sigma(a) x = \sum_{i=1}^k \sigma_i(a) x_i^2 - \sum_{i=1}^k \sum_{j=1}^k \sigma_i(a) x_i \sigma_j(a) x_j = \mathrm{E} X^2 - (\mathrm{E} X)^2 = \mathrm{Var} X \geq 0$. Straightforward algebra now gives the relative entropy $L_\sigma(y, \widehat{y}) = \sum_{j=1}^k y_j \ln(y_j/\widehat{y}_j)$ as the matching loss function. (To allow $y_j = 0$ or $\hat{y}_j = 0$, we adopt the standard convention that $0 \ln 0 = 0 \ln(0/0) = 0$ and $y \ln(y/0) = \infty$ for $y > 0$.)                                                                    □

In the relative loss bound proofs we use the basic property [JW98, Ama85]

$$\Delta_\phi(\widehat{a}', a) = \Delta_\phi(\widehat{a}', \widehat{a}) + \Delta_\phi(\widehat{a}, a) + (\phi(\widehat{a}) - \phi(a)) \cdot (\widehat{a}' - \widehat{a}) \ . \tag{3}$$

This shows that our distances do not satisfy the triangle inequality. Usually they are not symmetrical, either.

## 3   THE GENERAL ADDITIVE ALGORITHM

We consider on-line learning algorithms that at time $t$ first receive an input $x_t \in \mathbf{R}^n$, then produce an output $\widehat{y}_t \in \mathbf{R}^k$, and finally receive as feedback the desired output $y_t \in \mathbf{R}^k$. To define the *general additive algorithm*, assume we are given a transfer function

$\phi: \mathbf{R}^k \to \mathbf{R}^k$ that has a convex potential function. (We will later use the matching loss as a performance measure.) We also require that all the desired outputs $y_t$ are in the range of $\phi$. The algorithm's predictions are now given by $\hat{y}_t = \phi(\Omega_t x_t)$ where $\Omega_t \in \mathbf{R}^{k \times n}$ is the algorithm's *weight matrix* at time $t$. To see how the weight matrix is updated, assume further we have a parameterization function $\psi: \mathbf{R}^n \to \mathbf{R}^n$ with a distance $\Delta_\psi$. The algorithm maintains $kn$ real-valued parameters. We denote by $\Theta_t$ the $k \times n$ matrix of the values of these parameters immediately before trial $t$. Futher, we denote by $\theta_{t,j}$ the $j$th row of $\Theta_t$, and by $\psi(\Theta_t)$ the matrix with $\psi(\theta_{t,j})$ as its $j$th row. Given initial parameter values $\Theta_1$ and a learning rate $\eta > 0$, we now define the *general additive* (GA) *algorithm* as the algorithm that repeats at each trial $t$ the following prediction and update steps.

**Prediction:** Upon recieving the instance $x_t$, give the prediction $\hat{y}_t = \phi(\psi(\Theta_t)x_t)$.

**Update:** For $j = 1, \ldots, k$, set $\theta_{t+1,j} = \theta_{t,j} - \eta(\hat{y}_{t,j} - y_{t,j})x_t$.

Note that (2) implies $\nabla_{\hat{a}} \Delta_\phi(\hat{a}, a)) = \phi(\hat{a}) - \phi(a)$, so this update indeed turns out to be the same as (1) when we recall that $L_\phi(y_t, \hat{y}_t) = \Delta_\phi(\Omega_t x_t, a_t)$ where $y_t = \phi(a_t)$.

The update can be motivated by an optimization problem given in terms of the loss and distance. Consider updating an old parameter matrix $\Theta$ into a new matrix $\tilde{\Theta}$ based on a single input $x$ and desired output $y$. A natural goal would be to minimize the loss $L_\phi(y, \phi(\psi(\tilde{\Theta})x))$. However, the algorithm must avoid losing too much of the information it has gained during the previous trials and stored in the form of the old parameter matrix $\Theta$. We thus set as the algorithm's goal to minimize the sum $\Delta_\psi(\Theta, \tilde{\Theta}) + \eta L_\phi(y, \phi(\psi(\tilde{\Theta})x))$ where $\eta > 0$ is a parameter regulating how fast the algorithm is willing to move its parameters. Under certain regularity assumptions, the update rule of the GA algorithm can be shown to approximately solve this minimization problem. For more discussion and examples in the special case of linear regression, see [KW97]. An interesting related idea is using all the previous examples in the update instead of just the last one. For work along these lines in the linear case see Vovk [Vov97] and Foster [Fos91].

## 4   RELATIVE LOSS BOUNDS

Consider a sequence $S = ((x_1, y_1), \ldots, (x_\ell, y_\ell))$ of training examples, and let $\text{Loss}_\phi(\text{GA}, S) = \sum_{t=1}^\ell L_\phi(y_t, \hat{y}_t)$ be the loss incurred by the general additive algorithm on this sequence when it always uses its current weights $\Omega_t$ for making the $t$th prediction $\hat{y}_t$. Similarly, let $\text{Loss}_\phi(\Omega, S) = \sum_{t=1}^\ell L_\phi(y_t, \phi(\Omega x_t))$ be the loss of a fixed predictor $\Omega$. Basically, our goal is to show that if some $\Omega$ achieves a small loss, then the algorithm is not doing much worse, regardless of how the sequence $S$ was generated. Making additional probabilistic assumptions allows such on-line loss bounds to be converted into more traditional results about generalization errors [KW97]. To give the bounds for $\text{Loss}_\phi(\text{GA}, S)$ in terms of $\text{Loss}_\phi(\Omega, S)$ we need some additional parameters. The first one is the distance $\Delta_\psi(\Theta_1, \Theta)$ where $\Omega = \psi(\Theta)$ and $\Theta_1$ is the initial parameter matrix of the GA algorithm (which can be arbitrary). The second one is defined by

$$b_{\mathcal{X}, \psi} = \sup \left\{ x^T \mathrm{D}\psi(\theta)x \mid \theta \in \mathbf{R}^n, x \in \mathcal{X} \right\}$$

where $\mathcal{X} = \{x_1, \ldots, x_\ell\}$ is the set of input vectors and $\mathrm{D}\psi(\theta)$ is the Jacobian with $(\mathrm{D}\psi(\theta))_{ij} = \partial \psi_i(\theta)/\partial \theta_j$. The value $b_{\mathcal{X}, \psi}$ can be interpreted as the maximum norm of

any input vector in a norm defined by the parameterization function $\psi$. In Example 3 below we show how $b_{\chi,\psi}$ can easily be evaluated when $\psi$ is a linear function or the softmax function. The third parameter $c_\phi$, defined as

$$c_\phi = \sup\left\{\frac{(y-\widehat{y})^2}{2L_\phi(y,\widehat{y})} \mid y,\widehat{y} \in \overline{V_\phi}\right\} \ ,$$

relates the matching loss function for the transfer function $\phi$ to the square loss. In Example 4 we evaluate this constant for linear functions, the softmax function, and the one-dimensional case.

**Example 3** Consider bounding the value $x^{\mathrm{T}}\mathrm{D}\sigma(\theta)x$ where $\sigma$ is the softmax function. As we saw in Example 2, this value is a variance of a random variable with the range $\{x_1,\ldots,x_n\}$. Hence, we have $b_{\chi,\sigma} \le \max_{x\in\chi}(\max_i x_i - \min_i x_i)^2/4 \le \max_{x\in\chi}\|x\|_\infty^2$ where $\|x\|_\infty = \max_i |x_i|$.

If $\psi$ is a linear function with $\psi(\theta) = A\theta$ for a symmetrical positive definite $A$, we clearly have $b_{\chi,\psi} \le \lambda_{\max} \max_{x\in\chi} x^2$ where $\lambda_{\max}$ is the largest eigenvalue of $A$.                      □

**Example 4** For the softmax function $\sigma$ the matching loss function $L_\sigma$ is the relative entropy (see Example 2), for which it is well known that $L_\sigma(y,\widehat{y}) \ge 2(y-\widehat{y})^2$. Hence, we have $c_\phi \le 1/4$.

If $\phi$ is a linear function given by a symmetrical positive semidefinite matrix $A$, we see from Example 1 that $c_\phi$ is the largest eigenvalue of $A$.

Finally, in the special case $k = 1$, with $\phi\colon \mathbf{R} \to \mathbf{R}$ differentiable and strictly increasing, we can show $c_\phi \le Z$ if $Z$ is a bound such that $0 < \phi'(z) \le Z$ holds for all $z$.        □

Assume now we are given constants $b \ge b_{\chi,\psi}$ and $c \ge c_\phi$. Our first loss bound states that for any parameter matrix $\Theta$ we have

$$\mathrm{Loss}_\phi(\mathrm{GA}, S) \le 2\mathrm{Loss}_\phi(\psi(\Theta), S) + 4bc\Delta_\psi(\Theta_1, \Theta)$$

when the learning rate is chosen as $\eta = 1/(2bc)$. (Proofs are omitted from this extended abstract.) The advantage of this bound is that with a fixed learning rate it holds for any $\Theta$, so we need no advance knowledge about a good $\Theta$. The drawback is the factor 2 in front of $\mathrm{Loss}_\phi(\psi(\Theta), S)$, which suggests that asymptotically the algorithm might not ever achieve the performance of the best fixed predictor. A tighter bound can be achieved by more careful tuning. Thus, given constants $K \ge 0$ and $R > 0$, if we choose the learning rate as $\eta = (\sqrt{(bcR)^2 + KbcR} - bcR)/(Kbc)$ (with $\eta = 1/(2bc)$ if $K = 0$) we obtain

$$\mathrm{Loss}_\phi(\mathrm{GA}, S) \le \mathrm{Loss}_\phi(\psi(\Theta), S) + 2\sqrt{KbcR} + 4bcR$$

for any $\Theta$ that satisfies $\mathrm{Loss}_\phi(\psi(\Theta), S) \le K$ and $\Delta_\psi(\Theta_1, \Theta) \le R$. This shows that if we restrict our comparison to parameter matrices within a given distance $R$ of the initial matrix of the algorithm, and we have a reasonably good guess $K$ as to the loss of the best fixed predictor within this distance, this knowledge allows the algorithm to asymptotically match the performance of this best fixed predictor.

**Acknowledgments**

The authors thank Katy Azoury, Chris Bishop, Nicolò Cesa-Bianchi, David Helmbold, and Nick Littlestone for helpful discussions. Jyrki Kivinen was supported by the Academy of Finland and the ESPRIT project NeuroCOLT. Manfred Warmuth was supported by the NSF grant CCR 9700201.

# References

[Ama85]  S. Amari. *Differential Geometrical Methods in Statistics*. Springer Verlag, Berlin, 1985.

[AHW95]  P. Auer, M. Herbster, and M. K. Warmuth. Exponentially many local minima for single neurons. In *Proc. 1995 Neural Information Processing Conference*, pages 316–317. MIT Press, Cambridge, MA, November 1995.

[AW97]  K. Azoury and M. K. Warmuth. Relative loss bounds and the exponential family of distributions. Unpublished manuscript, 1997.

[Bud93]  M. Budinich. Some notes on perceptron learning. *J. Phys. A.: Math. Gen.*, 26:4237–4247, 1993.

[CB97]  N. Cesa-Bianchi. Analysis of two gradient-based algorithms for on-line regression. In *Proc. 10th Annu. Conf. on Comput. Learning Theory*, pages 163–170. ACM, 1997.

[Fos91]  D. P. Foster. Prediction in the worst case. *The Annals of Statistics*, 19(2):1084–1090, 1991.

[GLS97]  A. J. Grove, N. Littlestone, and D. Schuurmans. General convergence results for linear discriminant updates. In *Proc. 10th Annu. Conf. on Comput. Learning Theory*, pages 171–183. ACM, 1997.

[HKW95]  D. P. Helmbold, J. Kivinen, and M. K. Warmuth. Worst-case loss bounds for sigmoided linear neurons. In *Proc. Neural Information Processing Systems 1995*, pages 309–315. MIT Press, Cambridge, MA, November 1995.

[JW98]  A. K. Jagota and M. K. Warmuth. Continuous versus discrete-time nonlinear gradient descent: Relative loss bounds and convergence. Presented at *Fifth Symposium on Artificial Intelligence and Mathematics*, Ft. Lauderdale, FL, 1998.

[KW97]  J. Kivinen and M. K. Warmuth. Additive versus exponentiated gradient updates for linear prediction. *Information and Computation*, 132(1):1–64, January 1997.

[MN89]  P. McCullagh and J. A. Nelder. *Generalized Linear Models*. Chapman & Hall, New York, 1989.

[Vov97]  V. Vovk. Competitive on-line linear regression. In *Proc. Neural Information Processing Systems 1997*. MIT Press, Cambridge, MA, 1998.